# Moreau-Yosida Regularization for Grouped Tree Structure Learning

**Jun Liu**
Computer Science and Engineering
Arizona State University
J.Liu@asu.edu

**Jieping Ye**
Computer Science and Engineering
Arizona State University
Jieping.Ye@asu.edu

## Abstract

We consider the tree structured group Lasso where the structure over the features can be represented as a tree with leaf nodes as features and internal nodes as clusters of the features. The structured regularization with a pre-defined tree structure is based on a group-Lasso penalty, where one group is defined for each node in the tree. Such a regularization can help uncover the structured sparsity, which is desirable for applications with some meaningful tree structures on the features. However, the tree structured group Lasso is challenging to solve due to the complex regularization. In this paper, we develop an efficient algorithm for the tree structured group Lasso. One of the key steps in the proposed algorithm is to solve the Moreau-Yosida regularization associated with the grouped tree structure. The main technical contributions of this paper include (1) we show that the associated Moreau-Yosida regularization admits an analytical solution, and (2) we develop an efficient algorithm for determining the effective interval for the regularization parameter. Our experimental results on the AR and JAFFE face data sets demonstrate the efficiency and effectiveness of the proposed algorithm.

## 1 Introduction

Many machine learning algorithms can be formulated as a penalized optimization problem:

$$\min_{\mathbf{x}} l(\mathbf{x}) + \lambda\phi(\mathbf{x}), \tag{1}$$

where $l(\mathbf{x})$ is the empirical loss function (e.g., the least squares loss and the logistic loss), $\lambda > 0$ is the regularization parameter, and $\phi(\mathbf{x})$ is the penalty term. Recently, sparse learning via $\ell_1$ regularization [20] and its various extensions has received increasing attention in many areas including machine learning, signal processing, and statistics. In particular, the group Lasso [1, 16, 22] utilizes the group information of the features, and yields a solution with grouped sparsity. The traditional group Lasso assumes that the groups are non-overlapping. However, in many applications the features may form more complex overlapping groups. Zhao *et al.* [23] extended the group Lasso to the case of overlapping groups, imposing hierarchical relationships for the features. Jacob et al. [6] considered group Lasso with overlaps, and studied theoretical properties of the estimator. Jenatton et al. [7] considered the consistency property of the structured overlapping group Lasso, and designed an active set algorithm.

In many applications, the features can naturally be represented using certain tree structures. For example, the image pixels of the face image shown in Figure 1 can be represented as a tree, where each parent node contains a series of child nodes that enjoy spatial locality; genes/proteins may form certain hierarchical tree structures. Kim and Xing [9] studied the tree structured group Lasso for multi-task learning, where multiple related tasks follow a tree structure. One challenge in the practical application of the tree structured group Lasso is that the resulting optimization problem is much more difficult to solve than Lasso and group Lasso, due to the complex regularization.

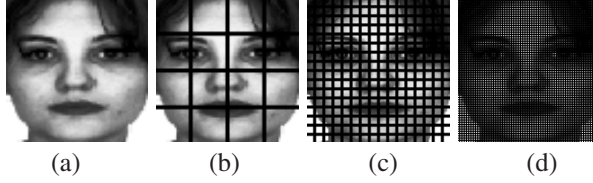

| (a) | (b) | (c) | (d) |

Figure 1: Illustration of the tree structure of a two-dimensional face image. The $64 \times 64$ image (a) can be divided into 16 sub-images in (b) according to the spatial locality, where the sub-images can be viewed as the child nodes of (a). Similarly, each $16 \times 16$ sub-image in (b) can be divided into 16 sub-images in (c), and such a process is repeated for the sub-images in (c) to get (d).

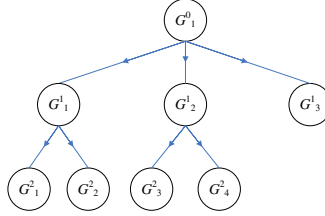

Figure 2: A sample index tree for illustration. Root: $G_1^0 = \{1, 2, 3, 4, 5, 6, 7, 8\}$. Depth 1: $G_1^1 = \{1, 2\}$, $G_2^1 = \{3, 4, 5, 6\}, G_3^1 = \{7, 8\}$. Depth 2: $G_1^2 = \{1\}, G_2^2 = \{2\}, G_3^2 = \{3, 4\}, G_4^2 = \{5, 6\}$.

In this paper, we develop an efficient algorithm for the tree structured group Lasso, i.e., the optimization problem (1) with $\phi(\cdot)$ being the grouped tree structure regularization (see Equation 2). One of the key steps in the proposed algorithm is to solve the Moreau-Yosida regularization [17, 21] associated with the grouped tree structure. The main technical contributions of this paper include: (1) we show that the associated Moreau-Yosida regularization admits an analytical solution, and the resulting algorithm for the tree structured group Lasso has a time complexity comparable to Lasso and group Lasso, and (2) we develop an efficient algorithm for determining the effective interval for the parameter $\lambda$, which is important in the practical application of the algorithm. We have performed experimental studies using the AR and JAFFE face data sets, where the features form a hierarchical tree structure based on the spatial locality as shown in Figure 1. Our experimental results demonstrate the efficiency and effectiveness of the proposed algorithm. Note that while the present paper was under review, we became aware of a recent work by Jenatton et al. [8] which applied block coordinate ascent in the dual and showed that the algorithm converges in one pass.

## 2 Grouped Tree Structure Regularization

We begin with the definition of the so-called index tree:

**Definition 1.** *For an index tree $T$ of depth $d$, we let $T_i = \{G_1^i, G_2^i, \ldots, G_{n_i}^i\}$ contain all the node(s) corresponding to depth $i$, where $n_0 = 1$, $G_1^0 = \{1, 2, \ldots, p\}$ and $n_i \geq 1, i = 1, 2, \ldots, d$. The nodes satisfy the following conditions: 1) the nodes from the same depth level have non-overlapping indices, i.e., $G_j^i \cap G_k^i = \emptyset, \forall i = 1, \ldots, d, j \neq k, 1 \leq j, k \leq n_i$; and 2) let $G_{j0}^{i-1}$ be the parent node of a non-root node $G_j^i$, then $G_j^i \subseteq G_{j0}^{i-1}$.*

Figure 2 shows a sample index tree. We can observe that 1) the index sets from different nodes may overlap, e.g., any parent node overlaps with its child nodes; 2) the nodes from the same depth level do not overlap; and 3) the index set of a child node is a subset of that of its parent node.

The grouped tree structure regularization is defined as:

$$\phi(\mathbf{x}) = \sum_{i=0}^{d} \sum_{j=1}^{n_i} w_j^i \|\mathbf{x}_{G_j^i}\|, \tag{2}$$

where $\mathbf{x} \in \mathbb{R}^p$, $w_j^i \geq 0$ $(i = 0, 1, \ldots, d, j = 1, 2, \ldots, n_i)$ is the pre-defined weight for the node $G_j^i$, $\| \cdot \|$ is the Euclidean norm, and $\mathbf{x}_{G_j^i}$ is a vector composed of the entries of $\mathbf{x}$ with the indices in $G_j^i$.

In the next section, we study the Moreau-Yosida regularization [17, 21] associated with (2), develop an analytical solution for such a regularization, propose an efficient algorithm for solving (1), and specify the meaningful interval for the regularization parameter $\lambda$.

# 3 Moreau-Yosida Regularization of $\phi(\cdot)$

The Moreau-Yosida regularization associated with the grouped tree structure regularization $\phi(\cdot)$ for a given $\mathbf{v} \in \mathbb{R}^p$ is given by:

$$\phi_\lambda(\mathbf{v}) = \min_{\mathbf{x}} \left\{ f(\mathbf{x}) = \frac{1}{2}\|\mathbf{x} - \mathbf{v}\|^2 + \lambda \sum_{i=0}^{d} \sum_{j=1}^{n_i} w_j^i \|\mathbf{x}_{G_j^i}\| \right\}, \tag{3}$$

for some $\lambda > 0$. Denote the minimizer of (3) as $\pi_\lambda(\mathbf{v})$. The Moreau-Yosida regularization has many useful properties: 1) $\phi_\lambda(\cdot)$ is continuously differentiable despite the fact that $\phi(\cdot)$ is non-smooth; 2) $\pi_\lambda(\cdot)$ is a non-expansive operator. More properties on the general Moreau-Yosida regularization can be found in [5, 10]. Note that, $f(\cdot)$ in (3) is indeed a special case of the problem (1) with $l(\mathbf{x}) = \frac{1}{2}\|\mathbf{x} - \mathbf{v}\|^2$. Our recent study has shown that, the efficient optimization of the Moreau-Yosida regularization is key to many optimization algorithms [13, Section 2]. Next, we focus on the efficient optimization of (3). For convenience of subsequent discussion, we denote $\lambda_j^i = \lambda w_j^i$.

## 3.1 An Analytical Solution

We show that the minimization of (3) admits an analytical solution. We first present the detailed procedure for finding the minimizer in Algorithm 1.

---

**Algorithm 1** Moreau-Yosida Regularization of the tree structured group Lasso (MY$_{\text{tgLasso}}$)

---

**Input:** $\mathbf{v} \in \mathbb{R}^p$, the index tree $T$ with nodes $G_j^i$ ($i = 0, 1, \ldots, d, j = 1, 2, \ldots, n_i$) that satisfy Definition 1, the weights $w_j^i \geq 0$ ($i = 0, 1, \ldots, d, j = 1, 2, \ldots, n_i$), $\lambda > 0$, and $\lambda_j^i = \lambda w_j^i$

**Output:** $\mathbf{u}^0 \in \mathbb{R}^p$

1: Set
$$\mathbf{u}^{d+1} = \mathbf{v}, \tag{4}$$

2: **for** $i = d$ to $0$ **do**
3:     **for** $j = 1$ to $n_i$ **do**
4:         Compute

$$\mathbf{u}_{G_j^i}^i = \begin{cases} \mathbf{0} & \|\mathbf{u}_{G_j^i}^{i+1}\| \leq \lambda_j^i \\ \frac{\|\mathbf{u}_{G_j^i}^{i+1}\| - \lambda_j^i}{\|\mathbf{u}_{G_j^i}^{i+1}\|} \mathbf{u}_{G_j^i}^{i+1} & \|\mathbf{u}_{G_j^i}^{i+1}\| > \lambda_j^i, \end{cases} \tag{5}$$

5:     **end for**
6: **end for**

---

In the implementation of the MY$_{\text{tgLasso}}$ algorithm, we only need to maintain a working variable $\mathbf{u}$, which is initialized with $\mathbf{v}$. We then traverse the index tree $T$ in the reverse breadth-first order to update $\mathbf{u}$. At the traversed node $G_j^i$, we update $\mathbf{u}_{G_j^i}$ according to the operation in (5), which reduces the Euclidean norm of $\mathbf{u}_{G_j^i}$ by at most $\lambda_j^i$. The time complexity of MY$_{\text{tgLasso}}$ is $O(\sum_{i=0}^{d} \sum_{j=1}^{n_i} |G_j^i|)$. By using Definition 1, we have $\sum_{j=1}^{n_i} |G_j^i| \leq p$. Therefore, the time complexity of MY$_{\text{tgLasso}}$ is $O(pd)$. If the tree is balanced, i.e., $d = O(\log p)$, then the time complexity of MY$_{\text{tgLasso}}$ is $O(p \log p)$.

MY$_{\text{tgLasso}}$ can help explain why the structured group sparsity can be induced. Let us analyze the tree given in Figure 2, with the solution denoted as $\mathbf{x}^*$. We let $w_j^i = 1, \forall i, j$, $\lambda = \sqrt{2}$, and $\mathbf{v} = [1, 2, 1, 1, 4, 4, 1, 1]^T$. After traversing the nodes of depth 2, we can get that the elements of $\mathbf{x}^*$ with indices in $G_1^2$ and $G_3^2$ are zero; and when the traversal continues to the nodes of depth 1, the elements of $\mathbf{x}^*$ with indices in $G_1^1$ and $G_3^1$ are set to zero, but those with $G_4^2$ are still nonzero. Finally, after traversing the root node, we obtain $\mathbf{x}^* = [0, 0, 0, 0, 1, 1, 0, 0]^T$.

Next, we show that $\mathrm{MY}_{\mathrm{tgLasso}}$ finds the exact minimizer of (3). The main result is summarized in the following theorem:

**Theorem 1.** $\mathbf{u}^0$ *returned by Algorithm 1 is the unique solution to* (3).

Before giving the detailed proof for Theorem 1, we introduce some notations, and present several technical lemmas.

Define the mapping $\phi_j^i : \mathbb{R}^p \to \mathbb{R}$ as

$$\phi_j^i(\mathbf{x}) = \|\mathbf{x}_{G_j^i}\|. \tag{6}$$

We can then express $\phi(\mathbf{x})$ defined in (2) as:

$$\phi(\mathbf{x}) = \sum_{i=0}^{d} \sum_{j=1}^{n_i} \lambda_j^i \phi_j^i(\mathbf{x}).$$

The subdifferential of $f(\cdot)$ defined in (3) at the point $\mathbf{x}$ can be written as:

$$\partial f(\mathbf{x}) = \mathbf{x} - \mathbf{v} + \sum_{i=0}^{d} \sum_{j=1}^{n_i} \lambda_j^i \partial \phi_j^i(\mathbf{x}), \tag{7}$$

where

$$\partial \phi_j^i(\mathbf{x}) = \begin{cases} \left\{ \mathbf{y} \in \mathbb{R}^p : \|\mathbf{y}\| \leq 1, \mathbf{y}_{\overline{G_j^i}} = \mathbf{0} \right\} & \text{if } \mathbf{x}_{G_j^i} = \mathbf{0} \\ \left\{ \mathbf{y} \in \mathbb{R}^p : \mathbf{y}_{G_j^i} = \frac{\mathbf{x}_{G_j^i}}{\|\mathbf{x}_{G_j^i}\|}, \mathbf{y}_{\overline{G_j^i}} = \mathbf{0} \right\} & \text{if } \mathbf{x}_{G_j^i} \neq \mathbf{0}, \end{cases} \tag{8}$$

and $\overline{G_j^i}$ denotes the complementary set of $G_j^i$.

**Lemma 1.** *For any* $1 \leq i \leq d, 1 \leq j \leq n_i$, *we can find a unique path from the node* $G_j^i$ *to the root node* $G_1^0$. *Let the nodes on this path be* $G_{r_l}^l$, *for* $l = 0, 1, \ldots, i$ *with* $r_0 = 1$ *and* $r_i = j$. *We have*

$$G_j^i \subseteq G_{r_l}^l, \forall l = 0, 1, \ldots, i-1. \tag{9}$$

$$G_j^i \cap G_r^l = \emptyset, \forall r \neq r_l, l = 1, 2, \ldots, i-1, r = 1, 2, \ldots, n_i. \tag{10}$$

**Proof**: According to Definition 1, we can find a unique path from the node $G_j^i$ to the root node $G_1^0$. In addition, based on the structure of the index tree, we have (9) and (10). $\quad\square$

**Lemma 2.** *For any* $i = 1, 2, \ldots, d, j = 1, 2, \ldots, n_i$, *we have*

$$\mathbf{u}_{G_j^i}^i \in \mathbf{u}_{G_j^i}^{i+1} - \lambda_j^i \left( \partial \phi_j^i(\mathbf{u}^i) \right)_{G_j^i}, \tag{11}$$

$$\partial \phi_j^i(\mathbf{u}^i) \subseteq \partial \phi_j^i(\mathbf{u}^0). \tag{12}$$

**Proof**: We can verify (11) using (5), (6) and (8).

For (12), it follows from (6) and (8) that, it is sufficient to verify that

$$\mathbf{u}_{G_j^i}^0 = \alpha_j^i \mathbf{u}_{G_j^i}^i, \text{ for some } \alpha_j^i \geq 0. \tag{13}$$

It follows from Lemma 1 that we can find a unique path from $G_j^i$ to $G_1^0$. Denote the nodes on the path as: $G_{r_l}^l$, where $l = 0, 1, \ldots, i$, $r_i = j$, and $r_0 = 1$. We first analyze the relationship between $\mathbf{u}_{G_j^i}^i$ and $\mathbf{u}_{G_j^i}^{i-1}$. If $\left\| \mathbf{u}_{G_{r_{i-1}}^{i-1}}^i \right\| \leq \lambda_{r_{i-1}}^{i-1}$, we have $\mathbf{u}_{G_{r_{i-1}}^{i-1}}^{i-1} = \mathbf{0}$, which leads to $\mathbf{u}_{G_j^i}^{i-1} = \mathbf{0}$ by using (9). Otherwise, if $\left\| \mathbf{u}_{G_{r_{i-1}}^{i-1}}^i \right\| > \lambda_{r_{i-1}}^{i-1}$, we have $\mathbf{u}_{G_{r_{i-1}}^{i-1}}^{i-1} = \frac{\left\| \mathbf{u}_{G_{r_{i-1}}^{i-1}}^i \right\| - \lambda_{r_{i-1}}^{i-1}}{\left\| \mathbf{u}_{G_{r_{i-1}}^{i-1}}^i \right\|} \mathbf{u}_{G_{r_{i-1}}^{i-1}}^i$, which leads to $\mathbf{u}_{G_j^i}^{i-1} = \frac{\left\| \mathbf{u}_{G_{r_{i-1}}^{i-1}}^i \right\| - \lambda_{r_{i-1}}^{i-1}}{\left\| \mathbf{u}_{G_{r_{i-1}}^{i-1}}^i \right\|} \mathbf{u}_{G_j^i}^i$ by using (9). Therefore, we have

$$\mathbf{u}_{G_j^i}^{i-1} = \beta_i \mathbf{u}_{G_j^i}^i, \text{ for some } \beta_i \geq 0. \tag{14}$$

By a similar argument, we have

$$\mathbf{u}_{G_{r_l}^l}^{l-1} = \beta_l \mathbf{u}_{G_{r_l}^l}^l, \beta_l \geq 0, \forall l = 1, 2, \dots, i-1. \tag{15}$$

Together with (9), we have

$$\mathbf{u}_{G_j^i}^{l-1} = \beta_l \mathbf{u}_{G_j^i}^l, \beta_l \geq 0, , \forall l = 1, 2, \dots, i-1. \tag{16}$$

From (14) and (16), we show (13) holds with $\alpha_j^i = \Pi_{l=1}^i \beta_l$. This completes the proof. $\quad\square$

We are now ready to prove our main result:

**Proof of Theorem 1**: It is easy to verify that $f(\cdot)$ defined in (3) is strongly convex, thus it admits a unique minimizer. Our methodology for the proof is to show that

$$\mathbf{0} \in \partial f(\mathbf{u}^0), \tag{17}$$

which is the sufficient and necessary condition for $\mathbf{u}^0$ to be the minimizer of $f(\cdot)$.

According to Definition 1, the leaf nodes are non-overlapping. We assume that the union of the leaf nodes equals to $\{1, 2, \dots, p\}$; otherwise, we can add to the index tree the additional leaf nodes with weight 0 to satisfy the aforementioned assumption. Clearly, the original index tree and the new index tree with the additional leaf nodes of weight 0 yield the same penalty $\phi(\cdot)$ in (2), the same Moreau-Yosida regularization in (3), and the same solution from Algorithm 1. Therefore, to prove (17), it suffices to show $\mathbf{0} \in \partial f(\mathbf{u}^0)_{G_j^i}$, for all the leaf nodes $G_j^i$. Next, we focus on establishing the following relationship:

$$\mathbf{0} \in \partial f(\mathbf{u}^0)_{G_1^d}. \tag{18}$$

It follows from Lemma 1 that, we can find a unique path from the node $G_1^d$ to the root $G_1^0$. Let the nodes on this path are $G_{r_l}^l$, for $l = 0, 1, \dots, d$ with $r_0 = 1$ and $r_d = 1$. By using (10) of Lemma 1, we can get that the nodes that contain the index set $G_1^d$ are exactly on the aforementioned path. In other words, $\forall \mathbf{x}$, we have

$$\left(\partial \phi_r^l(\mathbf{x})\right)_{G_1^d} = \{\mathbf{0}\}, \forall r \neq r_l, l = 1, 2, \dots, d-1, r = 1, 2, \dots, n_i \tag{19}$$

by using (6) and (8).

Applying (11) and (12) of Lemma 2 to each node on the aformetioned path, we have

$$\mathbf{u}_{G_{r_l}^l}^{l+1} - \mathbf{u}_{G_{r_l}^l}^l \in \lambda_{r_l}^l \left(\partial \phi_{r_l}^l(\mathbf{u}^l)\right)_{G_{r_l}^l} \subseteq \lambda_{r_l}^l \left(\partial \phi_{r_l}^l(\mathbf{u}^0)\right)_{G_{r_l}^l}, \forall l = 0, 1, \dots, d. \tag{20}$$

Making using of (9), we obtain from (20) the following relationship:

$$\mathbf{u}_{G_1^d}^{l+1} - \mathbf{u}_{G_1^d}^l \in \lambda_{r_l}^l \left(\partial \phi_{r_l}^l(\mathbf{u}^0)\right)_{G_1^d}, \forall l = 0, 1, \dots, d. \tag{21}$$

Adding (21) for $l = 0, 1, \dots, d$, we have

$$\mathbf{u}_{G_1^d}^{d+1} - \mathbf{u}_{G_1^d}^0 \in \sum_{l=0}^d \lambda_{r_l}^l \left(\partial \phi_{r_l}^l(\mathbf{u}^0)\right)_{G_1^d} \tag{22}$$

It follows from (4), (7), (19) and (22) that (18) holds.

Similarly, we have $\mathbf{0} \in f(\mathbf{u}^0)_{G_j^i}$ for the other leaf nodes $G_j^i$. Thus, we have (17). $\quad\square$

## 3.2 The Proposed Optimization Algorithm

With the analytical solution for $\pi_\lambda(\cdot)$, the minimizer of (3), we can apply many existing methods for solving (1). First, we show in the following lemma that, the optimal solution to (1) can be computed as a fixed point. We shall show in Section 3.3 that, the result in this lemma can also help determine the interval for the values of $\lambda$.

**Lemma 3.** *Let $\mathbf{x}^*$ be an optimal solution to* (1). *Then, $\mathbf{x}^*$ satisfies:*

$$\mathbf{x}^* = \pi_{\lambda\tau}(\mathbf{x}^* - \tau l'(\mathbf{x}^*)), \forall \tau > 0. \tag{23}$$

**Proof**: $\mathbf{x}^*$ is an optimal solution to (1), if and only if

$$0 \in l'(\mathbf{x}^*) + \lambda \partial \phi(\mathbf{x}^*), \tag{24}$$

which leads to

$$0 \in \mathbf{x}^* - (\mathbf{x}^* - \tau l'(\mathbf{x}^*)) + \lambda \tau \partial \phi(\mathbf{x}^*), \forall \tau > 0. \tag{25}$$

Thus, we have $\mathbf{x}^* = \arg\min_{\mathbf{x}} \frac{1}{2}\|\mathbf{x} - (\mathbf{x}^* - \tau l'(\mathbf{x}^*))\|^2 + \lambda \tau \phi(\mathbf{x})$. Recall that $\pi_\lambda(\cdot)$ is the minimizer of (3). We have (23). □

It follows from Lemma 3 that we can apply the fixed point continuation method [4] for solving (1). It is interesting to note that, with an appropriately chosen $\tau$, the scheme in (23) indeed corresponds to the gradient method developed for the composite function optimization [2, 19], achieving the global convergence rate of $O(1/k)$ for $k$ iterations. In addition, the scheme in (23) can be accelerated to obtain the accelerated gradient descent [2, 19], where the Moreau-Yosidea regularization also needs to be evaluated in each of its iteration. We employ the accelerated gradient descent developed in [2] for the optimization in this paper. The algorithm is called "tgLasso", which stands for the tree structured group Lasso. Note that, tgLasso includes our previous algorithm [11] as a special case, when the index tree is of depth 1 and $w_1^0 = 0$.

### 3.3 The Effective Interval for the Values of $\lambda$

When estimating the model parameters via (1), a key issue is to choose the appropriate values for the regularization parameter $\lambda$. A commonly used approach is to select the regularization parameter from a set of candidate values, whose values, however, need to be pre-specified in advance. Therefore, it is essential to specify the effective interval for the values of $\lambda$. An analysis of $\mathrm{MY}_{\mathrm{tgLasso}}$ in Algorithm 1 shows that, with increasing $\lambda$, the entries of the solution to (3) are monotonically decreasing. Intuitively, the solution to (3) shall be exactly zero if $\lambda$ is sufficiently large and all the entries of $\mathbf{x}$ are penalized in $\phi(\mathbf{x})$. Next, we summarize the main results of this subsection.

**Theorem 2.** *The zero point is a solution to* (1) *if and only if the zero point is a solution to* (3) *with* $\mathbf{v} = -l'(\mathbf{0})$. *For the penalty* $\phi(\mathbf{x})$, *let us assume that all entries of* $\mathbf{x}$ *are penalized, i.e.,* $\forall l \in \{1, 2, \ldots, p\}$, *there exists at least one node* $G_j^i$ *that contains* $l$ *and meanwhile* $w_j^i > 0$. *Then, for any* $0 < \|-l'(\mathbf{0})\| < +\infty$, *there exists a unique* $\lambda_{\max} < +\infty$ *satisfying: 1) if* $\lambda \geq \lambda_{\max}$ *the zero point is a solution to* (1), *and 2) if* $0 < \lambda < \lambda_{\max}$, *the zero point is not a solution to* (1).

**Proof**: If $\mathbf{x}^* = \mathbf{0}$ is the solution to (1), we have (24). Setting $\tau = 1$ in (23), we obtain that $\mathbf{x}^* = \mathbf{0}$ is also the solution to (3) with $\mathbf{v} = -l'(\mathbf{0})$. If $\mathbf{x}^* = \mathbf{0}$ is the solution to (3) with $\mathbf{v} = -l'(\mathbf{0})$, we have $\mathbf{0} \in l'(\mathbf{0}) + \lambda \partial \phi(\mathbf{0})$, which indicates that $\mathbf{x}^* = \mathbf{0}$ is the solution to (1).

The function $\phi(\mathbf{x})$ is closed convex. According to [18, Chapater 3.1.5], $\partial \phi(\mathbf{0})$ is a closed convex and non-empty bounded set. From (8), it is clear that $\mathbf{0} \in \partial \phi(\mathbf{0})$. Therefore, we have $\|\mathbf{x}\| \leq R, \forall \mathbf{x} \in \partial \phi(\mathbf{0})$, where $R$ is a finite radius constant. Let

$$S = \{\mathbf{x} : \mathbf{x} = -\alpha R l'(\mathbf{0})/\|l'(\mathbf{0})\|, \alpha \in [0, 1]\}$$

be the line segment from $\mathbf{0}$ to $-Rl'(\mathbf{0})/\|l'(\mathbf{0})\|$. It is obvious that $S$ is closed convex and bounded. Define $I = S \bigcap \partial \phi(\mathbf{0})$, which is clearly closed convex and bounded. Define

$$\tilde{\lambda}_{\max} = \|l'(\mathbf{0})\| / \max_{\mathbf{x} \in I} \|\mathbf{x}\|.$$

It follows from $\|l'(\mathbf{0})\| > 0$ and the boundedness of $I$ that $\tilde{\lambda}_{\max} > 0$. We first show $\tilde{\lambda}_{\max} < +\infty$. Otherwise, we have $I = \{\mathbf{0}\}$. Thus, $\forall \lambda > 0$, we have $-l'(\mathbf{0})/\lambda \notin \partial \phi(\mathbf{0})$, which indicates that $\mathbf{0}$ is neither the solution to (1) nor (3) with $\mathbf{v} = -l'(\mathbf{0})$. Recall the assumption that, $\forall l \in \{1, 2, \ldots, p\}$, there exists at least one node $G_j^i$ that contains $l$ and meanwhile $w_j^i > 0$. It follows from Algorithm 1 that, there exists a $\tilde{\lambda} < +\infty$ such that when $\lambda > \tilde{\lambda}$, $\mathbf{0}$ is a solution to (3) with $\mathbf{v} = -l'(\mathbf{0})$, leading to a contradiction. Therefore, we have $0 < \tilde{\lambda}_{\max} < +\infty$. Let $\lambda_{\max} = \tilde{\lambda}_{\max}$. The arguments hold since 1) if $\lambda \geq \lambda_{\max}$, then $-l'(\mathbf{0})/\lambda \in I \subseteq \partial \phi(\mathbf{0})$; and 2) if $0 < \lambda < \lambda_{\max}$, then $-l'(\mathbf{0})/\lambda \notin \partial \phi(\mathbf{0})$. □

When $l'(\mathbf{0}) = \mathbf{0}$, the problem (1) has a trivial zero solution. We next focus on the nontrivial case $l'(\mathbf{0}) \neq \mathbf{0}$. We present the algorithm for efficiently solving $\lambda_{\max}$ in Algorithm 2. In Step 1, $\lambda_0$ is an initial guess of the solution. Our empirical study shows that $\lambda_0 = \sqrt{\frac{\|l'(\mathbf{0})\|^2}{\sum_{i=0}^d \sum_{j=1}^{n_i}(w_j^i)^2}}$ works quite well. In Step 2-6, we specify an interval $[\lambda_1, \lambda_2]$ in which $\lambda_{\max}$ resides. Finally, in Step 7-14, we apply bisection for computing $\lambda_{\max}$.

**Algorithm 2** Finding $\lambda_{\max}$ via Bisection

---

**Input:** $l'(\mathbf{0})$, the index tree $T$ with nodes $G_j^i$ $(i = 0, 1, \ldots, d, j = 1, 2, \ldots, n_i)$, the weights $w_j^i \geq 0$
   $(i = 0, 1, \ldots, d, j = 1, 2, \ldots, n_i)$, $\lambda_0$, and $\delta = 10^{-10}$
**Output:** $\lambda_{\max}$
 1: Set $\lambda = \lambda_0$
 2: **if** $\phi_\lambda(-l'(\mathbf{0})) = \mathbf{0}$ **then**
 3:    Set $\lambda_2 = \lambda$, and find the largest $\lambda_1 = 2^{-i}\lambda, i = 1, 2, \ldots$ such that $\pi_{\lambda_1}(-l'(\mathbf{0})) \neq \mathbf{0}$
 4: **else**
 5:    Set $\lambda_1 = \lambda$, and find the smallest $\lambda_2 = 2^i\lambda, i = 1, 2, \ldots$ such that $\pi_{\lambda_2}(-l'(\mathbf{0})) = \mathbf{0}$
 6: **end if**
 7: **while** $\lambda_2 - \lambda_1 \geq \delta$ **do**
 8:    Set $\lambda = \frac{\lambda_1 + \lambda_2}{2}$
 9:    **if** $\pi_\lambda(-l'(\mathbf{0})) = \mathbf{0}$ **then**
10:       Set $\lambda_2 = \lambda$
11:    **else**
12:       Set $\lambda_1 = \lambda$
13:    **end if**
14: **end while**
15: $\lambda_{\max} = \lambda$

---

## 4   Experiments

We have conducted experiments to evaluate the efficiency and effectiveness of the proposed tgLasso algorithm on the face data sets JAFFE [14] and AR [15]. JAFFE contains 213 images of ten Japanese actresses with seven facial expressions: neutral, happy, disgust, fear, anger, sadness, and suprise. We used a subsect of AR that contains 400 images corresponding to 100 subjects, with each subject containing four facial expression: neutral, smile, anger, and scream. For both data sets, we resize the image size to $64 \times 64$, and make use of the tree structure depicted in Figure 1. Our task is to discriminate each facial expression from the rest ones. Thus, we have seven and four binary classification tasks for JAFFE and AR, respectively. We employ the least squares loss for $l(\cdot)$, and set the regularization parameter $\lambda = r \times \lambda_{\max}$, where $\lambda_{\max}$ is computed using Algorithm 2, and $r = \{5 \times 10^{-1}, 2 \times 10^{-1}, 1 \times 10^{-1}, 5 \times 10^{-2}, 2 \times 10^{-2}, 1 \times 10^{-2}, 5 \times 10^{-3}, 2 \times 10^{-3}\}$. The source codes, included in the SLEP package [12], are available online[1].

Table 1: Computational time (seconds) for one binary classification task (averaged over 7 and 4 runs for JAFFE and AR, respectively). The total time for all eight regularization parameters is reported.

|                          | JAFFE | AR   |
|--------------------------|-------|------|
| tgLasso                  | 30    | 73   |
| alternating algorithm [9]| 4054  | 5155 |

**Efficiency of the Proposed tgLasso**   We compare our proposed tgLasso with the recently proposed alternating algorithm [9] designed for the tree-guided group Lasso. We report the total computational time (seconds) for running one binary classification task (averaged over 7 and 4 tasks for JAFFE and AR, respectively) corresponding to the eight regularization parameters in Table 1. We can obseve that tgLasso is much more efficient than the alternating algorithm. We note that, the key step of tgLasso in each iteration is the associated Moreau-Yosida regularization, which can be efficiently computed due to the existence of an analytical solution; and the key step of the alternating algorithm in each iteration is the matrix inversion, which does not scale well to high-dimensional data.

**Classification Performance**   We compare the classification performance of tgLasso with Lasso. On AR, we use 50 subjects for training, and the remaining 50 subjects for testing; and on JAFFE, we use 8 subjects for training, and the remaining 2 subjects for testing. This subject-independent setting is challenging, as the subjects to be tested are not included in the training set. The reported results are averaged over 10 runs for randomly chosen subjects. For each binary classification task, we compute the balanced error rate [3] to cope with the unbalanced positive and negative samples. We

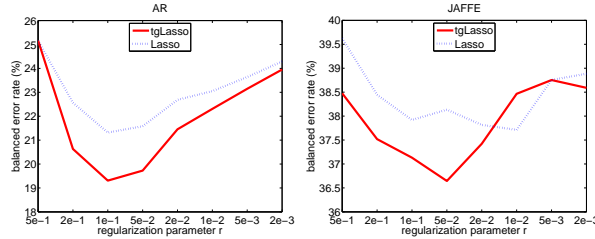

Figure 3: Classification performance comparison between Lasso and the tree structured group Lasso. The horizontal axis corresponds to different regularization parameters $\lambda = r \times \lambda_{\max}$.

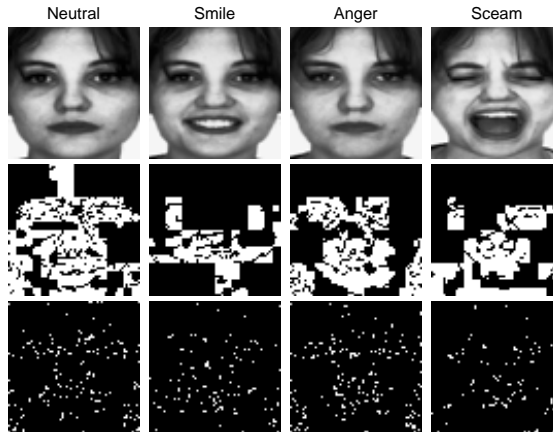

Figure 4: Markers obtained by Lasso, and tree structured group Lasso (white pixels correspond to the markers). First row: face images of four expression from the AR data set; Second row: the markers identified by tree structured group Lasso; Third row: the markers identified by Lasso.

report the averaged results in Figure 3. Results show that tgLasso outperforms Lasso in both cases. This verifies the effectiveness of tgLasso in incorporating the tree structure in the formulation, i.e., the spatial locality information of the face images. Figure 4 shows the markers identified by tgLasso and Lasso under the best regularization parameter. We can observe from the figure that tgLasso results in a block sparsity solution, and most of the selected pixels are around mouths and eyes.

## 5   Conclusion

In this paper, we consider the efficient optimization for the tree structured group Lasso. Our main technical result show the Moreau-Yosida regularization associated with the tree structured group Lasso admits an analytical solution. Based on the Moreau-Yosida regularization, we an design efficient algorithm for solving the grouped tree structure regularized optimization problem for smooth convex loss functions, and develop an efficient algorithm for determining the effective interval for the parameter $\lambda$. Our experimental results on the AR and JAFFE face data sets demonstrate the efficiency and effectiveness of the proposed algorithm. We plan to apply the proposed algorithm to other applications in computer vision and bioinformatics involving the tree structure.

**Acknowledgments**

This work was supported by NSF IIS-0612069, IIS-0812551, CCF-0811790, IIS-0953662, NGA HM1582-08-1-0016, NSFC 60905035, 61035003, and the Office of the Director of National Intelligence (ODNI), Intelligence Advanced Research Projects Activity (IARPA), through the US Army.

## Footnotes

[1] http://www.public.asu.edu/~jye02/Software/SLEP/

# References

[1] F. Bach, G. Lanckriet, and M. Jordan. Multiple kernel learning, conic duality, and the SMO algorithm. In *International conference on Machine learning*, 2004.

[2] A. Beck and M. Teboulle. A fast iterative shrinkage-thresholding algorithm for linear inverse problems. *SIAM Journal on Imaging Sciences*, 2(1):183–202, 2009.

[3] I. Guyon, A. B. Hur, S. Gunn, and G. Dror. Result analysis of the nips 2003 feature selection challenge. In *Neural Information Processing Systems*, pages 545–552, 2004.

[4] E.T. Hale, W. Yin, and Y. Zhang. Fixed-point continuation for $\ell_1$-minimization: Methodology and convergence. *SIAM Journal on Optimization*, 19(3):1107–1130, 2008.

[5] J. Hiriart-Urruty and C. Lemaréchal. *Convex Analysis and Minimization Algorithms I & II*. Springer Verlag, Berlin, 1993.

[6] L. Jacob, G. Obozinski, and J. Vert. Group lasso with overlap and graph lasso. In *International Conference on Machine Learning*, 2009.

[7] R. Jenatton, J.-Y. Audibert, and F. Bach. Structured variable selection with sparsity-inducing norms. Technical report, arXiv:0904.3523v2, 2009.

[8] R. Jenatton, J. Mairal, G. Obozinski, and F. Bach. Proximal methods for sparse hierarchical dictionary learning. In *International Conference on Machine Learning*, 2010.

[9] S. Kim and E. P. Xing. Tree-guided group lasso for multi-task regression with structured sparsity. In *International Conference on Machine Learning*, 2010.

[10] C. Lemaréchal and C. Sagastizábal. Practical aspects of the Moreau-Yosida regularization I: Theoretical properties. *SIAM Journal on Optimization*, 7(2):367–385, 1997.

[11] J. Liu, S. Ji, and J. Ye. Multi-task feature learning via efficient $\ell_{2,1}$-norm minimization. In *Uncertainty in Artificial Intelligence*, 2009.

[12] J. Liu, S. Ji, and J. Ye. *SLEP: Sparse Learning with Efficient Projections*. Arizona State University, 2009.

[13] J. Liu, L. Yuan, and J. Ye. An efficient algorithm for a class of fused lasso problems. In *ACM SIGKDD Conference on Knowledge Discovery and Data Mining*, 2010.

[14] M. J. Lyons, J. Budynek, and S. Akamatsu. Automatic classification of single facial images. *IEEE Transactions on Pattern Analysis and Machine Intelligence*, 21(12):1357–1362, 1999.

[15] A.M. Martinez and R. Benavente. The AR face database. Technical report, 1998.

[16] L. Meier, S. Geer, and P. Bühlmann. The group lasso for logistic regression. *Journal of the Royal Statistical Society: Series B*, 70:53–71, 2008.

[17] J.-J. Moreau. Proximité et dualité dans un espace hilbertien. *Bulletin de la Societe mathematique de France*, 93:273–299, 1965.

[18] Y. Nesterov. *Introductory Lectures on Convex Optimization: A Basic Course*. Kluwer Academic Publishers, 2004.

[19] Y. Nesterov. Gradient methods for minimizing composite objective function. *CORE Discussion Paper*, 2007.

[20] R. Tibshirani. Regression shrinkage and selection via the lasso. *Journal of the Royal Statistical Society Series B*, 58(1):267–288, 1996.

[21] K. Yosida. *Functional Analysis*. Springer Verlag, Berlin, 1964.

[22] M. Yuan and Y. Lin. Model selection and estimation in regression with grouped variables. *Journal Of The Royal Statistical Society Series B*, 68(1):49–67, 2006.

[23] P. Zhao, G. Rocha, and B. Yu. The composite absolute penalties family for grouped and hierarchical variable selection. *Annals of Statistics*, 37(6A):3468–3497, 2009.

